# Learning a Distance Metric from Relative Comparisons

**Matthew Schultz and Thorsten Joachims**
Department of Computer Science
Cornell University
Ithaca, NY 14853
{schultz,tj}@cs.cornell.edu

## Abstract

This paper presents a method for learning a distance metric from relative comparison such as "A is closer to B than A is to C". Taking a Support Vector Machine (SVM) approach, we develop an algorithm that provides a flexible way of describing qualitative training data as a set of constraints. We show that such constraints lead to a convex quadratic programming problem that can be solved by adapting standard methods for SVM training. We empirically evaluate the performance and the modelling flexibility of the algorithm on a collection of text documents.

## 1 Introduction

Distance metrics are an essential component in many applications ranging from supervised learning and clustering to product recommendations and document browsing. Since designing such metrics by hand is difficult, we explore the problem of learning a metric from examples. In particular, we consider relative and qualitative examples of the form "A is closer to B than A is to C". We believe that feedback of this type is more easily available in many application setting than quantitative examples (e.g. "the distance between A and B is 7.35") as considered in metric Multidimensional Scaling (MDS) (see [4]), or absolute qualitative feedback (e.g. "A and B are similar", "A and C are not similar") as considered in [11].

Building on the study in [7], search-engine query logs are one example where feedback of the form "A is closer to B than A is to C" is readily available for learning a (more semantic) similarity metric on documents. Given a ranked result list for a query, documents that are clicked on can be assumed to be semantically closer than those documents that the user observed but decided to not click on (i.e. "$A_{click}$ is closer to $B_{click}$ than $A_{click}$ is to $C_{noclick}$"). In contrast, drawing the conclusion that "$A_{click}$ and $C_{noclick}$ are not similar" is probably less justified, since a $C_{noclick}$ high in the presented ranking is probably still closer to $A_{click}$ than most documents in the collection.

In this paper, we present an algorithm that can learn a distance metric from such relative and qualitative examples. Given a parametrized family of distance metrics, the algorithms discriminately searches for the parameters that best fulfill the training examples. Taking a maximum-margin approach [9], we formulate the training problem as a convex quadratic

program for the case of learning a weighting of the dimensions. We evaluate the performance and the modelling flexibility of the algorithm on a collection of text documents.

The notation used throughout this paper is as follows. Vectors are denoted with an arrow $\vec{x}$ where $x_i$ is the $i^{th}$ entry in vector $\vec{x}$. The vector $\vec{0}$ is the vector composed of all zeros, and $\vec{1}$ is the vector composed of all ones. $\vec{x}^T$ is the transpose of vector $\vec{x}$ and the dot product is denoted by $\vec{x}^T \vec{y}$. We denote the element-wise product of two vectors $\vec{x} = (x_1, ..., x_n)^T$ and $\vec{y} = (y_1, ..., y_n)^T$ as $\vec{x} * \vec{y} = (x_1 y_1, ..., x_n y_n)^T$.

## 2 Learning from Relative Qualitative Feedback

We consider the following learning setting. Given is a set $X_{train}$ of objects $\vec{x}_i \in \Re^N$. As training data, we receive a subset $P_{train}$ of all potential relative comparisons defined over the set $X_{train}$. Each relative comparison $(i, j, k) \in P_{train}$ with $\vec{x}_i, \vec{x}_j, \vec{x}_k \in X_{train}$ has the semantic

$$\vec{x}_i \text{ is closer to } \vec{x}_j \text{ than } \vec{x}_i \text{ is to } \vec{x}_k.$$

The goal of the learner is to learn a weighted distance metric $d_{\vec{w}}(\cdot, \cdot)$ from $P_{train}$ and $X_{train}$ that best approximates the desired notion of distance on a new set of test points $X_{test}$, $X_{train} \cap X_{test} = \emptyset$. We evaluate the performance of a metric $d_{\vec{w}}(\cdot, \cdot)$ by how many relative comparisons $P_{test}$ it fulfills on the test set.

## 3 Parameterized Distance Metrics

A (pseudo) distance metric $d(\vec{x}, \vec{y})$ is a function over pairs of objects $\vec{x}$ and $\vec{y}$ from some set X. $d(\vec{x}, \vec{y})$ is a pseudo metric, iff it obeys the four following properties for all $\vec{x}, \vec{y}$, and $\vec{z}$:

$$d(\vec{x}, \vec{x}) = 0, \quad d(\vec{x}, \vec{y}) = d(\vec{y}, \vec{x}), \quad d(\vec{x}, \vec{y}) \geq 0, \quad d(\vec{x}, \vec{y}) + d(\vec{y}, \vec{z}) \geq d(\vec{x}, \vec{z})$$

It is a metric, iff it also obeys $d(\vec{x}, \vec{y}) = 0 \Rightarrow \vec{x} = \vec{y}$.

In this paper, we consider a distance metric $d_{A,W}(\vec{x}, \vec{y})$ between vectors $\vec{x}, \vec{y} \in \Re^N$ parameterized by two matrices, $A$ and $W$.

$$d_{A,W}(\vec{x}, \vec{y}) = \sqrt{(\vec{x} - \vec{y})^T A W A^T (\vec{x} - \vec{y})} \tag{1}$$

$W$ is a diagonal matrix with non-negative entries and $A$ is any real matrix. Note that the matrix $AWA^T$ is semi-positive definite so that $d_{A,W}(\vec{x}, \vec{y})$ is a valid distance metric.

This parametrization is very flexible. In the simplest case, A is the identity matrix, $I$, and $d_{I,W}(\vec{x}, \vec{y}) = \sqrt{(\vec{x} - \vec{y})^T I W I^T (\vec{x} - \vec{y})} = \sqrt{(\vec{x} - \vec{y})^T W (\vec{x} - \vec{y})}$ is a weighted, Euclidean distance $d_{I,W}(\vec{x}, \vec{y}) = \sqrt{\sum_i W_{ii}(x_i - y_i)^2}$.

In a general case, A can be any real matrix. This corresponds to applying a linear transformation to the input data with the matrix A. After the transformation, the distance becomes a Euclidean distance on the transformed input points $A^T \vec{x}$, $A^T \vec{y}$.

$$d_{A,W}(\vec{x}, \vec{y}) = \sqrt{((\vec{x} - \vec{y})^T A) W (A^T (\vec{x} - \vec{y}))} \tag{2}$$

The use of kernels $K(\vec{x}, \vec{y}) = \phi(\vec{x}) \phi(\vec{y})$ suggests a particular choice of A. Let $\Phi$ be the matrix where the i-th column is the (training) vector $\vec{x}_i$ projected into a feature space using

the function $\phi(\vec{x}_i)$. Then

$$d_{\Phi,W}(\phi(\vec{x}),\phi(\vec{y})) \quad = \quad \sqrt{((\phi(\vec{x})-\phi(\vec{y}))^T\Phi)W(\Phi^T(\phi(\vec{x})-\phi(\vec{y})))} \tag{3}$$

$$= \quad \sqrt{\sum_{i=1}^{n} W_{ii}(K(\vec{x},\vec{x}_i)-K(\vec{y},\vec{x}_i))^2} \tag{4}$$

is a distance metric in the feature space.

## 4   An SVM Algorithm for Learning from Relative Comparisons

Given a training set $P_{train}$ of $n$ relative comparisons over a set of vectors $X_{train}$, and the matrix $A$, we aim to fit the parameters in the diagonal matrix $W$ of distance metric $d_{A,W}(\vec{x},\vec{y})$ so that the training error (i.e. the number of violated constraints) is minimized. Finding a solution of zero training error is equivalent to finding a $W$ that fulfills the following set of constraints.

$$\forall(i,j,k) \in P_{train} : d_{A,W}(\vec{x_i},\vec{x_k}) - d_{A,W}(\vec{x_i},\vec{x_j}) > 0 \tag{5}$$

If the set of constraints is feasible and a $W$ exists that fulfills all constraints, the solution is typically not unique. We aim to select a matrix $AWA^T$ such that $d_{A,W}(\vec{x},\vec{y})$ remains as close to an unweighted Euclidean metric as possible. Following [8], we minimize the norm of the eigenvalues $||\Lambda||^2$ of $AWA^T$. Since $||\Lambda||^2 = ||AWA^T||_F^2$, this leads to the following optimization problem.

$$min \quad \frac{1}{2}||AWA^T||_F^2$$
$$s.t. \quad \forall(i,j,k)\in P_{train} : (\vec{x}_i-\vec{x}_k)^TAWA^T(\vec{x}_i-\vec{x}_k) - (\vec{x}_i-\vec{x}_j)^TAWA^T(\vec{x}_i-\vec{x}_j) \geq 1$$
$$W_{ii} \geq 0$$

Unlike in [8], this formulation ensures that $d_{A,W}(\vec{x},\vec{y})$ is a metric, avoiding the need for semi-definite programming like in [11].

As in classification SVMs, we add slack variables [3] to account for constraints that cannot be satisfied. This leads to the following optimization problem.

$$min \quad \frac{1}{2}||AWA^T||_F^2 + C\sum_{i,j,k}\xi_{ijk}$$
$$s.t. \quad \forall(i,j,k)\in P_{train} : (\vec{x}_i-\vec{x}_k)^TAWA^T(\vec{x}_i-\vec{x}_k) - (\vec{x}_i-\vec{x}_j)^TAWA^T(\vec{x}_i-\vec{x}_j) \geq 1-\xi_{ijk}$$
$$\xi_{ijk} \geq 0$$
$$W_{ii} \geq 0$$

The sum of the slack variables $\xi_{ijk}$ in the objective is an upper bound on the number of violated constraints.

All distances $d_{A,W}(\vec{x},\vec{y})$ can be written in the following linear form. If we let $\vec{w}$ be the diagonal elements of W then the distance $d_{A,W}$ can be written as

$$d_{A,W}(\vec{x},\vec{y}) \quad = \quad \sqrt{((\vec{x}-\vec{y})^TA)W(A^T(\vec{x}-\vec{y}))}$$

$$= \quad \sqrt{\vec{w}^T(A^T\vec{x}-A^T\vec{y})*(A^T\vec{x}-A^T\vec{y})} \tag{6}$$

where $*$ denotes the element-wise product. If we let $\vec{\Delta}^{x_i,x_j} = (A^T\vec{x}_i - A^T\vec{x_k})*(A^T\vec{x}_i - A^T\vec{x_k})$, then the constraints in the optimization problem can be rewritten in the following linear form.

$$\forall(i,j,k) \in P_{train} : \vec{w}^T(\vec{\Delta}^{x_i,x_k} - \vec{\Delta}^{x_i,x_k}) \geq 1-\xi_{ijk} \tag{7}$$

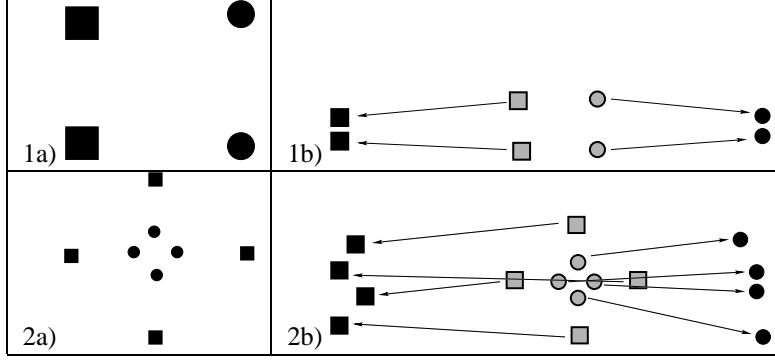

Figure 1: Graphical example of using different A matrices. In example 1, A is the identity matrix and in example 2 A is composed of the training examples projected into high dimensional space using an RBF kernel.

Furthermore, the objective function is quadratic, so that the optimization problem can be written as

$$min \qquad \frac{1}{2}\vec{w}^T L \vec{w} + C \sum_{i,j,k} \xi_{ijk}$$

$$s.t. \qquad \forall (i,j,k) \in P_{train} : \vec{w}^T(\vec{\Delta}^{x_i,x_k} - \vec{\Delta}^{x_i,x_j}) \geq 1 - \xi_{ijk}$$

$$\xi_{ijk} \geq 0$$

$$W_{ii} \geq 0 \qquad\qquad (8)$$

For the case of $A = I$, $||AWA^T||_F^2 = w^T L w$ with $L = I$. For the case of $A = \Phi$, we define $L = (A^T A) * (A^T A)$ so that $||AWA^T||_F^2 = w^T L w$. Note that $L$ is positive semi-definite in both cases and that, therefore, the optimization problem is convex quadratic.

## 5 Experiments

In Figure 1, we display a graphical example of our method. Example 1 is an example of a weighted Euclidean distance. The input data points are shown in 1a) and our training constraints specify that the distance between two square points should be less than the distance to a circle. Similarly, circles should be closer to each other than to squares. Figure 1 (1b) shows the points after an MDS analysis with the learned distance metric as input. This learned distance metric intuitively correponds to stretching the x-axis and shrinking the y-axis in the original input space.

Example 2 in Figure 1 is an example where we have a similar goal of grouping the squares together and separating them from the circles. In this example though, there is no way to use a linear weighting measure to accomplish this task. We used an RBF kernel and learned a distance metric to separate the clusters. The result is shown in 2b.

To validate the method using a real world example, we ran several experiments on the WEBKB data set [5]. In order to illustrate the versatility of relative comparisons, we generated three different distance metrics from the same data set and ran three types of tests: an accuracy test, a learning curve to show how the method generalizes from differing amounts of training data, and an MDS test to graphically illustrate the new distance measures.

The experimental setup for each of the experiments was the same. We first split $X$, the set of all 4,183 documents, into separate training and test sets, $X_{train}$ and $X_{test}$. 70% of the

all examples $X$ added to $X_{train}$ and the remaining 30% are in $X_{test}$. We used a binary feature vector without stemming or stop word removal (63,949 features) to represent each document because it is the least biased distance metric to start out with. It also performed best among several different variations of term weighting, stemming and stopword removal.

The relative comparison sets, $P_{train}$ and $P_{test}$, were generated as follows. We present results for learning three different notions of distance.

- *University Distance*: This distance is small when the two examples, $\vec{x}, \vec{y}$, are from the same university and larger otherwise. For this data set we used webpages from seven universities.

- *Topic Distance*: This distance metric is small when the two examples, $\vec{x}, \vec{y}$, are from the same topic (e.g. both are student webpages) and larger when they are each from a different topic. There are four topics: Student, Faculty, Course and Project webpages.

- *Topic+FacultyStudent Distance*: Again when two examples, $\vec{x}, \vec{y}$, are from the same topic then they have a small distance between them and a larger distance when they come from different topics. However, we add the additional constraint that the distance between a faculty and a student page is smaller than the distance to pages from other topics.

To build the training constraints, $P_{train}$, we first randomly selected three documents, $x_i, x_j, x_k$, from $X_{train}$. For the University Distance we added the triplet $(i, j, k)$ to $P_{train}$ if $x_i$ and $x_j$ were from the same university and $x_k$ was from a different university. In building $P_{train}$ for the Topic Distance we added the $(i, j, k)$ to $P_{train}$ if $x_i$ and $x_j$ were from the same topic (e.g. "Student Webpages") and $x_k$ was from a different topic (e.g. "Project Webpages"). For the Topic+FacultyStudent Distance, the training triple $(i, j, k)$ was added to $P_{train}$ if either the topic rule occurred, when $x_i$ and $x_j$ were from the same topic and $x_k$ was from a different topic, or if $x_i$ was a faculty webpage, $x_j$ was a student webpage and $x_k$ was either a project or course webpage. Thus the constraints would specify that a student webpage is closer to a faculty webpage than a faculty webpage is to a course webpage.

| | Learned $d_{\vec{w}}(\cdot, \cdot)$ | Binary | TFIDF |
|---|---|---|---|
| University Distance | 98.43% | 67.88% | 80.72% |
| Topic Distance | 75.40% | 61.82% | 55.57% |
| Topic+FacultyStudent Distance | 79.67% | 63.08% | 55.06% |

Table 1: Accuracy of different distance metrics on an unseen test set $P_{test}$.

The results of the learned distance measures on unseen test sets $P_{test}$ are reported in Table 1. In each experiment the regularization parameter $C$ was set to 1 and we used $A = I$. We report the percentage of the relative comparisons in $P_{test}$ that were satisfied for each of the three experiments. As a baseline for comparison, we give the results for the static (not learned) distance metric that performs best on the test set. The best performing metric for all static Euclidean distances (Binary and TFIDF) used stemming and stopword removal, which our learned distance did not use. The learned University Distance satisfied 98.43% of the constraints. This verifies that the learning method can effectively find the relevant features, since pages usually mentioned which university they were from. For the other distances, both the Topic Distance and Topic+FacultyStudent Distance satisfied more than 13% more constraints in $P_{test}$ than the best unweighted distance. Using a kernel instead of $A = I$ did not yield improved results.

For the second test, we illustrate on the Topic+FacultyStudent data set how the prediction accuracy of the method scales with the number of training constraints. The learning curve

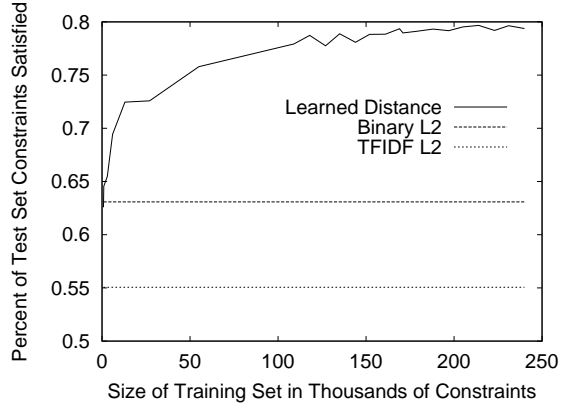

Figure 2: Learning curves for the Topic+FacultyStudent dataset where the x axis is the size of the training set $P_{train}$ plotted against the y axis which is the percent of constraints in $P_{test}$ that were satisfied.

is shown in Figure 2 where we plot the training set size (in number of constraints) versus the percentage of test constraints satisfied. The test set $P_{test}$ was held constant and sampled in the same way as the training set ($|P_{test}| = 85,907$). As Figure 2 illustrates, after the data set contained more than 150,000 constraints, the performance of the algorithm remained relatively constant.

As a final test of our method, we graphically display our distance metrics in Table 7. We plot three distance metrics: The standard binary distance (Figure a) for the Topic Distance, the learned metric for Topic Distance (Figure b) and, and the learned metric for the Topic+FacultyStudent Distance (Figure c). To produce the plots in Table 7, all pairwise distances between the points in $X_{test}$ were computed and then projected into 2D using a classical, metric MDS algorithm [1].

Figure a) in Table 7 is the result of using the pairwise distances resulting from the un-weighted, binary $L_2$ norm in MDS. There is no clear distinction between any of the clusters in 2 dimensions. In Figure b) we see the results of the learned Topic Distance measure. The classes were reasonably separated from each other. Figure c) shows the result of using the learned Topic+FacultyStudent Distance metric. When compared to Figure b), the Faculty and Student webpages have now moved closer together as desired.

## 6   Related Work

The most relevant related work is the work of Xing et al [11] which focused on the problem of learning a distance metric to increase the accuracy of nearest neighbor algorithms. Their work used absolute, qualitative feedback such as "A is similar to B" or "A is dissimilar to B" which is different from the relative constraints considered here. Secondly, their method does not use regularization.

Related are also techniques for semi-supervised clustering, as it is also considered in [11]. While [10] does not change the distance metric, [2] uses gradient descent to adapt a param-eterized distance metric according to user feedback.

Other related work are dimension reduction techniques such as Multidimensional Scaling (MDS) [4] and Latent Semantic Indexing [6]. Metric MDS techniques take as input a matrix D of dissimilarities (or similarities) between all points in some collection and then seeks to arrange the points in a d-dimensional space to minimize the stress. The stress of the

arrangement is roughly the difference between the distances in the d-dimensional space and the distances input in matrix D. LSI uses an eigenvalue decomposition of the original input space to find the first d principal eigenvectors to describe the data in d dimensions. Our work differs because the input is a set of relative comparisons, not quantitative distances and does not project the data into a lower dimensional space. Non-metric MDS is more similar to our technique than metric MDS. Instead of preserving the exact distances input, the non-metric MDS seeks to maintain the rank order of the distances. However, the goal of our method is not a low dimensional projection, but a new distance metric in the original space.

## 7    Conclusion and Future Work

In this paper we presented a method for learning a weighted Euclidean distance from relative constraints. This was accomplished by solving a convex optimization problem similar to SVMs to find the maximum margin weight vector. One of the main benefits of the algorithm is that the new type of the constraint enables its use in a wider range of applications than conventional methods. We evaluated the method on a collection of high dimensional text documents and showed that it can successfully learn different notions of distance.

Future work is needed both with respect to theory and application. In particular, we do not yet know generalization error bounds for this problem. Furthermore, the power of the method would be increased, if it was possible to learn more complex metrics that go beyond feature weighting, for example by incorporating kernels in a more adaptive way.

## References

[1] A. Buja, D. Swayne, M. Littman, and N. Dean. Xgvis: Interactive data visualization with multidimensional scaling. *Journal of Computational and Graphical Statistics*, to appear.

[2] D. Cohn, R. Caruana, and A. McCallum. Semi-supervised clustering with user feedback. Technical Report TR2003-1892, Cornell University, 2003.

[3] Corinna Cortes and Vladimir Vapnik. Support-vector networks. *Machine Learning*, 20(3):273–297, 1995.

[4] T. Cox and M. Cox. *Multidimensional Scaling*. Chapman & Hall, London, 1994.

[5] M. Craven, D. DiPasquo, D. Freitag, A. McCallum, T. Mitchell, K. Nigam, and S. Slattery. Learning to extract symbolic knowledge from the world wide web. *Proceedings of the 15th National Conference on Artificial Intelligence (AAAI-98)*, 1998.

[6] Scott C. Deerwester, Susan T. Dumais, Thomas K. Landauer, George W. Furnas, and Richard A. Harshman. Indexing by latent semantic analysis. *Journal of the American Society of Information Science*, 41(6):391–407, 1990.

[7] T. Joachims. Optimizing search engines using clickthrough data. *Proceedings of the ACM Conference on Knowledge Discovery and Data Mining (KDD)*, 2002.

[8] I.W. Tsang and J.T. Kwok. Distance metric learning with kernels. *Proceedings of the International Conference on Artificial Neural Networks*, 2003.

[9] V. Vapnik. *Statistical Learning Theory*. Wiley, Chichester, GB, 1998.

[10] Kiri Wagstaff, Claire Cardie, Seth Rogers, and Stefan Schroedl. Constrained K-means clustering with background knowledge. In *Proc. 18th International Conf. on Machine Learning*, pages 577–584. Morgan Kaufmann, San Francisco, CA, 2001.

[11] E.P. Xing, A.Y. Ng, M.I. Jordan, and S. Russell. Distance metric learning, with application to clustering with side information. *Advances in Neural Information Processing Systems*, 2002.

a)

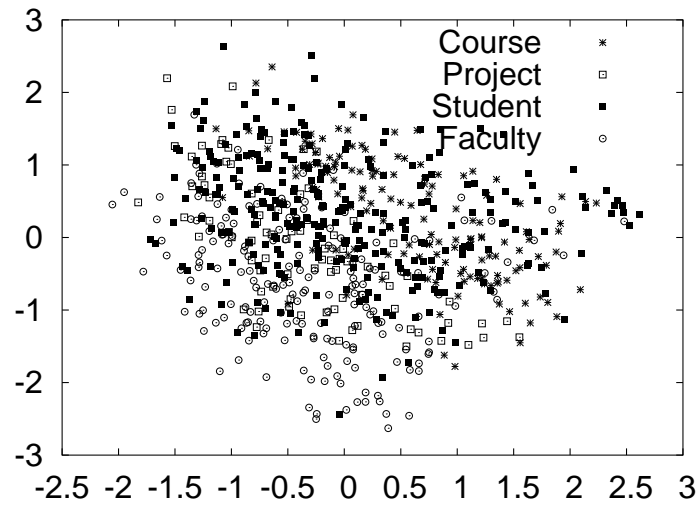

b)

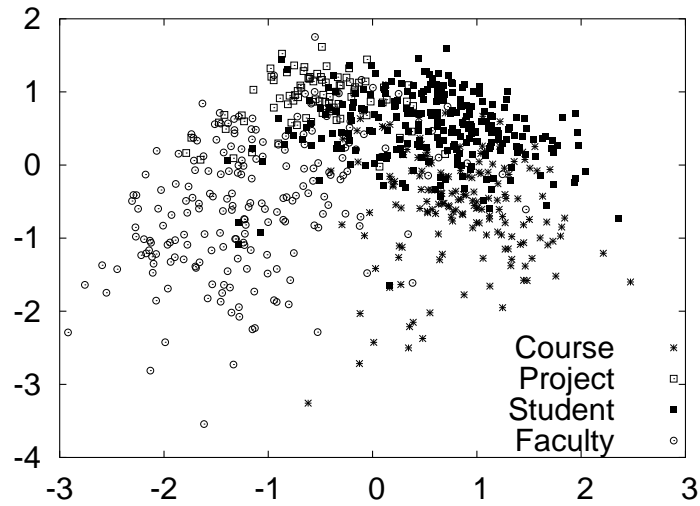

c)

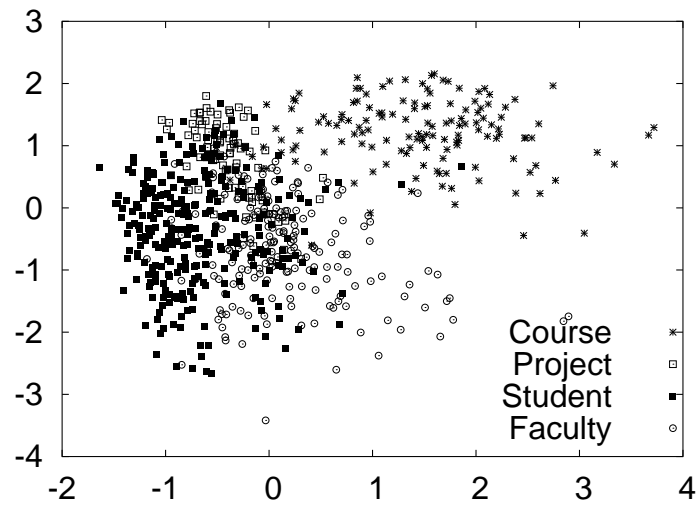

Table 2: MDS plots of distance functions: a) is the unweighted $L_2$ distance, b) is the Topic Distance, and c) is the Topic+FacultyStudent distance.